# A Better Way to Pretrain Deep Boltzmann Machines

**Ruslan Salakhutdinov**
Department of Statistics and Computer Science
University of Toronto
rsalakhu@cs.toronto.edu

**Geoffrey Hinton**
Department of Computer Science
University of Toronto
hinton@cs.toronto.edu

## Abstract

We describe how the pretraining algorithm for Deep Boltzmann Machines (DBMs) is related to the pretraining algorithm for Deep Belief Networks and we show that under certain conditions, the pretraining procedure improves the variational lower bound of a two-hidden-layer DBM. Based on this analysis, we develop a different method of pretraining DBMs that distributes the modelling work more evenly over the hidden layers. Our results on the MNIST and NORB datasets demonstrate that the new pretraining algorithm allows us to learn better generative models.

## 1 Introduction

A Deep Boltzmann Machine (DBM) is a type of binary pairwise Markov Random Field with multiple layers of hidden random variables. Maximum likelihood learning in DBMs, and other related models, is very difficult because of the hard inference problem induced by the partition function [3, 1, 12, 6]. Multiple layers of hidden units make learning in DBM's far more difficult [13]. Learning meaningful DBM models, particularly when modelling high-dimensional data, relies on the heuristic greedy pretraining procedure introduced by [7], which is based on learning a stack of modified Restricted Boltzmann Machines (RBMs). Unfortunately, unlike the pretraining algorithm for Deep Belief Networks (DBNs), the existing procedure lacks a proof that adding additional layers improves the variational bound on the log-probability that the model assigns to the training data.

In this paper, we first show that under certain conditions, the pretraining algorithm improves a variational lower bound of a two-layer DBM. This result gives a much deeper understanding of the relationship between the pretraining algorithms for Deep Boltzmann Machines and Deep Belief Networks. Using this understanding, we introduce a new pretraining procedure for DBMs and show that it allows us to learn better generative models of handwritten digits and 3D objects.

## 2 Deep Boltzmann Machines (DBMs)

A Deep Boltzmann Machine is a network of symmetrically coupled stochastic binary units. It contains a set of visible units $\mathbf{v} \in \{0, 1\}^D$, and a series of layers of hidden units $\mathbf{h}^{(1)} \in \{0, 1\}^{F_1}$, $\mathbf{h}^{(2)} \in \{0, 1\}^{F_2}$,..., $\mathbf{h}^{(L)} \in \{0, 1\}^{F_L}$. There are connections only between units in adjacent layers. Consider a DBM with three hidden layers, as shown in Fig. 1, left panel. The probability that the DBM assigns to a visible vector $\mathbf{v}$ is:

$$P(\mathbf{v}; \theta) = \frac{1}{\mathcal{Z}(\theta)} \sum_{\mathbf{h}} \exp \left( \sum_{ij} W_{ij}^{(1)} v_i h_j^{(1)} + \sum_{jl} W_{jl}^{(2)} h_j^{(1)} h_l^{(2)} + \sum_{lm} W_{lm}^{(3)} h_l^{(2)} h_m^{(3)} \right), \qquad (1)$$

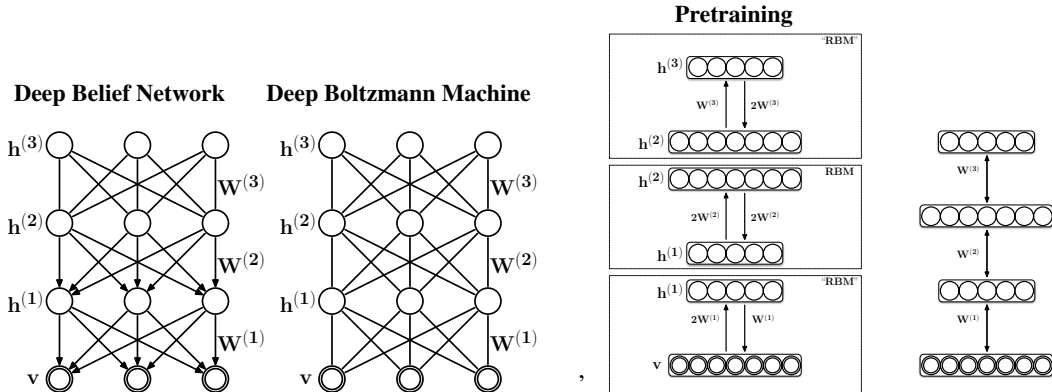

Figure 1: **Left:** Deep Belief Network (DBN) and Deep Boltzmann Machine (DBM). The top two layers of a DBN form an undirected graph and the remaining layers form a belief net with directed, top-down connections. For a DBM, all the connections are undirected. **Right** Pretraining a DBM with three hidden layers consists of learning a stack of RBMs that are then composed to create a DBM. The first and last RBMs in the stack need to be modified by using asymmetric weights.

where $\mathbf{h} = \{\mathbf{h}^{(1)}, \mathbf{h}^{(2)}, \mathbf{h}^{(3)}\}$ are the set of hidden units, and $\theta = \{\mathbf{W}^{(1)}, \mathbf{W}^{(2)}, \mathbf{W}^{(3)}\}$ are the model parameters, representing visible-to-hidden and hidden-to-hidden symmetric interaction terms[1]. Setting $\mathbf{W}^{(2)}$=0 and $\mathbf{W}^{(3)}$=0 recovers the Restricted Boltzmann Machine (RBM) model.

**Approximate Learning:** Exact maximum likelihood learning in this model is intractable, but efficient approximate learning of DBMs can be carried out by using a mean-field inference to estimate data-dependent expectations, and an MCMC based stochastic approximation procedure to approximate the model's expected sufficient statistics [7]. In particular, consider approximating the true posterior $P(\mathbf{h}|\mathbf{v};\theta)$ with a fully factorized approximating distribution over the three sets of hidden units: $Q(\mathbf{h}|\mathbf{v};\boldsymbol{\mu}) = \prod_{j=1}^{F_1} \prod_{l=1}^{F_2} \prod_{k=1}^{F_3} q(h_j^{(1)}|\mathbf{v}) q(h_l^{(2)}|\mathbf{v}) q(h_k^{(3)}|\mathbf{v})$, where $\boldsymbol{\mu} = \{\boldsymbol{\mu}^{(1)}, \boldsymbol{\mu}^{(2)}, \boldsymbol{\mu}^{(3)}\}$ are the mean-field parameters with $q(h_i^{(l)} = 1) = \mu_i^{(l)}$ for $l = 1, 2, 3$. In this case, we can write down the variational lower bound on the log-probability of the data, which takes a particularly simple form:

$$\log P(\mathbf{v};\theta) \geq \mathbf{v}^\top \mathbf{W}^{(1)} \boldsymbol{\mu}^{(1)} + {\boldsymbol{\mu}^{(1)}}^\top \mathbf{W}^{(2)} \boldsymbol{\mu}^{(2)} + {\boldsymbol{\mu}^{(2)}}^\top \mathbf{W}^{(3)} \boldsymbol{\mu}^{(3)} - \log \mathcal{Z}(\theta) + \mathcal{H}(Q), \quad (2)$$

where $\mathcal{H}(\cdot)$ is the entropy functional. Learning proceeds by finding the value of $\boldsymbol{\mu}$ that maximizes this lower bound for the current value of model parameters $\theta$, which results in a set of the mean-field fixed-point equations. Given the variational parameters $\boldsymbol{\mu}$, the model parameters $\theta$ are then updated to maximize the variational bound using stochastic approximation (for details see [7, 11, 14, 15]).

## 3 Pretraining Deep Boltzmann Machines

The above learning procedure works quite poorly when applied to DBMs that start with randomly initialized weights. Hidden units in higher layers are very under-constrained so there is no consistent learning signal for their weights. To alleviate this problem, [7] introduced a layer-wise pretraining algorithm based on learning a stack of "modified" Restricted Boltzmann Machines (RBMs).

The idea behind the pretraining algorithm is straightforward. When learning parameters of the first layer "RBM", the bottom-up weights are constrained to be twice the top-down weights (see Fig. 1, right panel). Intuitively, using twice the weights when inferring the states of the hidden units $\mathbf{h}^{(1)}$ compensates for the initial lack of top-down feedback. Conversely, when pretraining the last "RBM" in the stack, the top-down weights are constrained to be twice the bottom-up weights. For all the intermediate RBMs the weights are halved in both directions when composing them to form a DBM, as shown in Fig. 1, right panel.

This heuristic pretraining algorithm works surprisingly well in practice. However, it is solely motivated by the need to end up with a model that has symmetric weights, and does not provide any

useful insights into what is happening during the pretraining stage. Furthermore, unlike the pretraining algorithm for Deep Belief Networks (DBNs), it lacks a proof that each time a layer is added to the DBM, the variational bound improves.

## 3.1 Pretraining Algorithm for Deep Belief Networks

We first briefly review the pretraining algorithm for Deep Belief Networks [2], which will form the basis for developing a new pretraining algorithm for Deep Boltzmann Machines.

Consider pretraining a two-layer DBN using a stack of RBMs. After learning the first RBM in the stack, we can write the generative model as: $p(\mathbf{v}; \mathbf{W}^{(1)}) = \sum_{\mathbf{h}^{(1)}} p(\mathbf{v}|\mathbf{h}^{(1)}; \mathbf{W}^{(1)}) p(\mathbf{h}^{(1)}; \mathbf{W}^{(1)})$. The second RBM in the stack attempts to replace the prior $p(\mathbf{h}^{(1)}; \mathbf{W}^{(1)})$ by a better model $p(\mathbf{h}^{(1)}; \mathbf{W}^{(2)}) = \sum_{\mathbf{h}^{(2)}} p(\mathbf{h}^{(1)}, \mathbf{h}^{(2)}; \mathbf{W}^{(2)})$, thus improving the fit to the training data.

More formally, for any approximating distribution $Q(\mathbf{h}^{(1)}|\mathbf{v})$, the DBN's log-likelihood has the following variational lower bound on the log probability of the training data $\{\mathbf{v}_1, ..., \mathbf{v}_N\}$:

$$\sum_{n=1}^{N} \log P(\mathbf{v}_n) \geq \sum_n \mathrm{E}_{Q(\mathbf{h}^{(1)}|\mathbf{v}_n)} \left[ \log P(\mathbf{v}_n|\mathbf{h}^{(1)}; \mathbf{W}^{(1)}) \right] - \sum_n \mathrm{KL} \left( Q(\mathbf{h}^{(1)}|\mathbf{v}_n) || P(\mathbf{h}^{(1)}; \mathbf{W}^{(1)}) \right).$$

We set $Q(\mathbf{h}^{(1)}|\mathbf{v}_n; \mathbf{W}^{(1)}) = P(\mathbf{h}^{(1)}|\mathbf{v}_n; \mathbf{W}^{(1)})$, which is the true factorial posterior of the first layer RBM. Initially, when $\mathbf{W}^{(2)} = \mathbf{W}^{(1)\top}$, $Q(\mathbf{h}^{(1)}|\mathbf{v}_n)$ defines the DBN's true posterior over $\mathbf{h}^{(1)}$, and the bound is tight. Maximizing the bound with respect to $\mathbf{W}^{(2)}$ only affects the last KL term in the above equation, and amounts to maximizing:

$$\frac{1}{N} \sum_{n=1}^{N} \sum_{\mathbf{h}^{(1)}} Q(\mathbf{h}^{(1)}|\mathbf{v}_n; \mathbf{W}^{(1)}) P(\mathbf{h}^{(1)}; \mathbf{W}^{(2)}). \tag{3}$$

This is equivalent to training the second layer RBM with vectors drawn from $Q(\mathbf{h}^{(1)}|\mathbf{v}; \mathbf{W}^{(1)})$ as data. Hence, the second RBM in the stack learns a better model of the mixture over all $N$ training cases: $1/N \sum_n Q(\mathbf{h}^{(1)}|\mathbf{v}_n; \mathbf{W}^{(1)})$, called the "aggregated posterior". This scheme can be extended to training higher-layer RBMs.

Observe that during the pretraining stage the whole prior of the lower-layer RBM is replaced by the next RBM in the stack. This leads to the hybrid Deep Belief Network model, with the top two layers forming a Restricted Boltzmann Machine, and the lower layers forming a directed sigmoid belief network (see Fig. 1, left panel).

## 3.2 A Variational Bound for Pretraining a Two-layer Deep Boltzmann Machine

Consider a simple two-layer DBM with tied weights $\mathbf{W}^{(2)} = \mathbf{W}^{(1)\top}$, as shown in Fig. 2a:

$$P(\mathbf{v}; \mathbf{W}^{(1)}) = \frac{1}{\mathcal{Z}(\mathbf{W}^{(1)})} \sum_{\mathbf{h}^{(1)}, \mathbf{h}^{(2)}} \exp \left( \mathbf{v}^\top \mathbf{W}^{(1)} \mathbf{h}^{(1)} + \mathbf{h}^{(1)\top} \mathbf{W}^{(1)\top} \mathbf{h}^{(2)} \right). \tag{4}$$

Similar to DBNs, for any approximate posterior $Q(\mathbf{h}^{(1)}|\mathbf{v})$, we can write a variational lower bound on the log probability that this DBM assigns to the training data:

$$\sum_{n=1}^{N} \log P(\mathbf{v}_n) \geq \sum_n \mathrm{E}_{Q(\mathbf{h}^{(1)}|\mathbf{v}_n)} \left[ \log P(\mathbf{v}_n|\mathbf{h}^{(1)}; \mathbf{W}^{(1)}) \right] - \sum_n \mathrm{KL} \left( Q(\mathbf{h}^{(1)}|\mathbf{v}_n) || P(\mathbf{h}^{(1)}; \mathbf{W}^{(1)}) \right). \tag{5}$$

The key insight is to note that the model's marginal distribution over $\mathbf{h}^{(1)}$ is the product of two identical distributions, one defined by an RBM composed of $\mathbf{h}^{(1)}$ and $\mathbf{v}$, and the other defined by an identical RBM composed of $\mathbf{h}^{(1)}$ and $\mathbf{h}^{(2)}$ [8]:

$$P(\mathbf{h}^{(1)}; \mathbf{W}^{(1)}) = \frac{1}{\mathcal{Z}(\mathbf{W}^{(1)})} \underbrace{\left( \sum_{\mathbf{v}} e^{\mathbf{v}^\top \mathbf{W}^{(1)} \mathbf{h}^{(1)}} \right)}_{\text{RBM with } \mathbf{h}^{(1)} \text{ and } \mathbf{v}} \underbrace{\left( \sum_{\mathbf{h}^{(2)}} e^{\mathbf{h}^{(2)\top} \mathbf{W}^{(1)} \mathbf{h}^{(1)}} \right)}_{\text{RBM with } \mathbf{h}^{(1)} \text{ and } \mathbf{h}^{(2)}}. \tag{6}$$

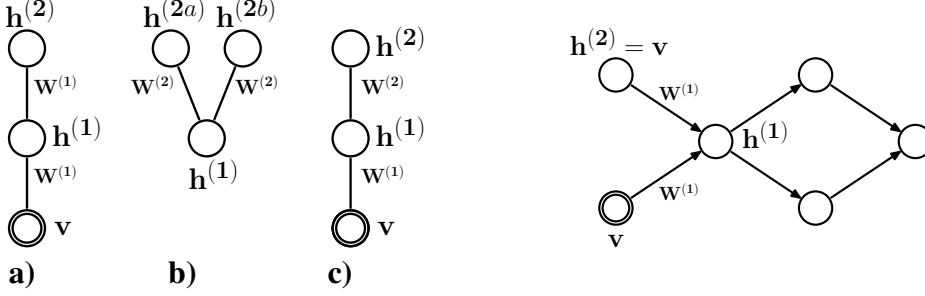

Figure 2: **Left:** Pretraining a Deep Boltzmann Machine with two hidden layers. **a)** The DBM with tied weights. **b)** The second RBM with two sets of replicated hidden units, which will replace half of the 1$^{\text{st}}$RBM's prior. **c)** The resulting DBM with modified second hidden layer. **Right:** The DBM with tied weights is trained to model the data using one-step contrastive divergence.

The idea is to keep one of these two RBMs and replace the other by the square root of a better prior $P(\mathbf{h}^{(1)}; \mathbf{W}^{(2)})$. In particular, another RBM with two sets of replicated hidden units and tied weights $P(\mathbf{h}^{(1)}; \mathbf{W}^{(2)}) = \sum_{\mathbf{h}^{(2a)}, \mathbf{h}^{(2b)}} P(\mathbf{h}^{(1)}, \mathbf{h}^{(2a)}, \mathbf{h}^{(2b)}; \mathbf{W}^{(2)})$ is trained to be a better model of the aggregated variational posterior $\frac{1}{N} \sum_n Q(\mathbf{h}^{(1)}|\mathbf{v}_n; \mathbf{W}^{(1)})$ of the first model (see Fig. 2b). By initializing $\mathbf{W}^{(2)} = \mathbf{W}^{(1)\top}$, the second-layer RBM has exactly the same prior over $\mathbf{h}^{(1)}$ as the original DBM. If the RBM is trained by maximizing the log likelihood objective:

$$\sum_n \sum_{\mathbf{h}^{(1)}} Q(\mathbf{h}^{(1)}|\mathbf{v}_n) \log P(\mathbf{h}^{(1)}; \mathbf{W}^{(2)}), \qquad (7)$$

then we obtain:

$$\sum_n \text{KL}(Q(\mathbf{h}^{(1)}|\mathbf{v}_n)||P(\mathbf{h}^{(1)}; \mathbf{W}^{(2)})) \leq \sum_n \text{KL}(Q(\mathbf{h}^{(1)}|\mathbf{v}_n)||P(\mathbf{h}^{(1)}; \mathbf{W}^{(1)})). \qquad (8)$$

Similar to Eq. 6, the distribution over $\mathbf{h}^{(1)}$ defined by the second-layer RBM is also the product of two identical distributions. Once the two RBMs are composed to form a two-layer DBM model (see Fig. 2c), the marginal distribution over $\mathbf{h}^{(1)}$ is the geometric mean of the two probability distributions: $P(\mathbf{h}^{(1)}; \mathbf{W}^{(1)}), P(\mathbf{h}^{(1)}; \mathbf{W}^{(2)})$ defined by the first and second-layer RBMs:

$$P(\mathbf{h}^{(1)}; \mathbf{W}^{(1)}, \mathbf{W}^{(2)}) = \frac{1}{\mathcal{Z}(\mathbf{W}^{(1)}, \mathbf{W}^{(2)})} \left( \sum_{\mathbf{v}} e^{\mathbf{v}^\top \mathbf{W}^{(1)} \mathbf{h}^{(1)}} \right) \left( \sum_{\mathbf{h}^{(2)}} e^{\mathbf{h}^{(1)\top} \mathbf{W}^{(2)} \mathbf{h}^{(2)}} \right). \qquad (9)$$

Based on Eqs. 8, 9, it is easy to show that the variational lower bound of Eq. 5 improves because replacing half of the prior by a better model reduces the KL divergence from the variational posterior:

$$\sum_n \text{KL} \left( Q(\mathbf{h}^{(1)}|\mathbf{v}_n)||P(\mathbf{h}^{(1)}; \mathbf{W}^{(1)}, \mathbf{W}^{(2)}) \right) \leq \sum_n \text{KL} \left( Q(\mathbf{h}^{(1)}|\mathbf{v}_n)||P(\mathbf{h}^{(1)}; \mathbf{W}^{(1)}) \right). \qquad (10)$$

Due to the convexity of asymmetric divergence, this is guaranteed to improve the variational bound of the training data by *at least half as much as fully replacing the original prior*.

This result highlights a major difference between DBNs and DBMs. The procedure for adding an extra layer to a DBN replaces the full prior over the previous top layer, whereas the procedure for adding an extra layer to a DBM only replaces half of the prior. So in a DBM, the weights of the bottom level RBM perform much more of the work than in a DBN, where the weights are only used to define the last stage of the generative process $P(\mathbf{v}|\mathbf{h}^{(1)}; \mathbf{W}^{(1)})$.

This result also suggests that adding layers to a DBM will give diminishing improvements in the variational bound, compared to adding layers to a DBN. This may explain why DBMs with three hidden layers typically perform worse than the DBMs with two hidden layers [7, 8]. On the other hand, the disadvantage of the pretraining procedure for Deep Belief Networks is that the top-layer RBM is forced to do most of the modelling work. This may also explain the need to use a large number of hidden units in the top-layer RBM [2].

There is, however, a way to design a new pretraining algorithm that would spread the modelling work more equally across all layers, hence bypassing shortcomings of the existing pretraining algorithms for DBNs and DBMs.

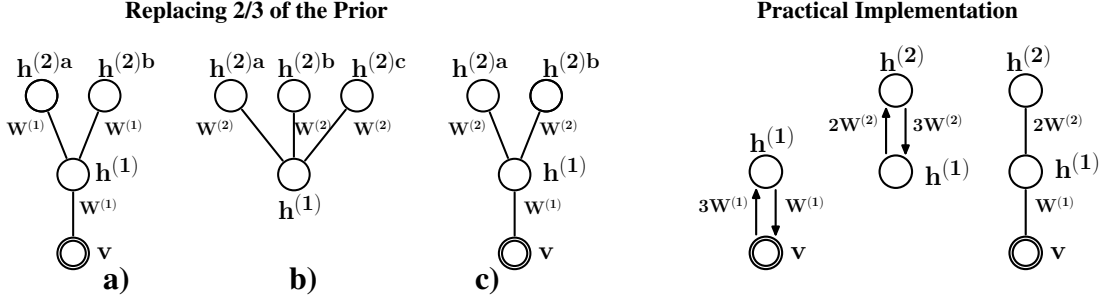

**Figure 3: Left:** Pretraining a Deep Boltzmann Machine with two hidden layers. **a)** The DBM with tied weights. **b)** The second layer RBM is trained to model $2/3$ of the $1^{\text{st}}$ RBM's prior. **c)** The resulting DBM with modified second hidden layer. **Right:** The corresponding practical implementation of the pretraining algorithm that uses asymmetric weights.

### 3.3 Controlling the Amount of Modelling Work done by Each Layer

Consider a slightly modified two-layer DBM with two groups of replicated $2^{\text{nd}}$-layer units, $\mathbf{h}^{(2a)}$ and $\mathbf{h}^{(2b)}$, and tied weights (see Fig. 3a). The model's marginal distribution over $\mathbf{h}^{(1)}$ is the product of three identical RBM distributions, defined by $\mathbf{h}^{(1)}$ and $\mathbf{v}$, $\mathbf{h}^{(1)}$ and $\mathbf{h}^{(2a)}$, and $\mathbf{h}^{(1)}$ and $\mathbf{h}^{(2b)}$:

$$P(\mathbf{h}^{(1)}; \mathbf{W}^{(1)}) = \frac{1}{\mathcal{Z}(\mathbf{W}^{(1)})} \left( \sum_{\mathbf{v}} e^{\mathbf{v}^\top \mathbf{W}^{(1)} \mathbf{h}^{(1)}} \right) \left( \sum_{\mathbf{h}^{(2a)}} e^{\mathbf{h}^{(2a)\top} \mathbf{W}^{(1)} \mathbf{h}^{(1)}} \right) \left( \sum_{\mathbf{h}^{(2b)}} e^{\mathbf{h}^{(2b)\top} \mathbf{W}^{(1)} \mathbf{h}^{(1)}} \right).$$

During the pretraining stage, we keep one of these RBMs and replace the other two by a better prior $P(\mathbf{h}^{(1)}; \mathbf{W}^{(2)})$. To do so, similar to Sec. 3.2, we train another RBM, but with three sets of hidden units and tied weights (see Fig. 3b). When we combine the two RBMs into a DBM, the marginal distribution over $\mathbf{h}^{(1)}$ is the geometric mean of three probability distributions: one defined by the first-layer RBM, and the remaining two defined by the second-layer RBMs:

$$P(\mathbf{h}^{(1)}; \mathbf{W}^{(1)}, \mathbf{W}^{(2)}) = \frac{1}{\mathcal{Z}(\mathbf{W}^{(1)}, \mathbf{W}^{(2)})} P(\mathbf{h}^{(1)}; \mathbf{W}^{(1)}) P(\mathbf{h}^{(1)}; \mathbf{W}^{(2)}) P(\mathbf{h}^{(1)}; \mathbf{W}^{(2)})$$

$$= \frac{1}{\mathcal{Z}(\mathbf{W}^{(1)}, \mathbf{W}^{(2)})} \left( \sum_{\mathbf{v}} e^{\mathbf{v}^\top \mathbf{W}^{(1)} \mathbf{h}^{(1)}} \right) \left( \sum_{\mathbf{h}^{(2a)}} e^{\mathbf{h}^{(2a)\top} \mathbf{W}^{(2)} \mathbf{h}^{(1)}} \right) \left( \sum_{\mathbf{h}^{(2b)}} e^{\mathbf{h}^{(2b)\top} \mathbf{W}^{(2)} \mathbf{h}^{(1)}} \right).$$

In this DBM, $2/3$ of the first RBM's prior over the first hidden layer has been replaced by the prior defined by the second-layer RBM. The variational bound on the training data is guaranteed to improve by at least $2/3$ as much as fully replacing the original prior. Hence in this slightly modified DBM model, the second layer performs $2/3$ of the modelling work compared to the first layer. Clearly, controlling the number of replicated hidden groups allows us to easily control the amount of modelling work left to the higher layers in the stack.

### 3.4 Practical Implementation

So far, we have made the assumption that we start with a two-layer DBM with tied weights. We now specify how one would train this initial set of tied weights $\mathbf{W}^{(1)}$.

Let us consider the original two-layer DBM in Fig. 2a with tied weights. If we knew the initial state vector $\mathbf{h}^{(2)}$, we could train this DBM using one-step contrastive divergence (CD) with mean field reconstructions of both the visible states $\mathbf{v}$ and the top-layer states $\mathbf{h}^{(2)}$, as shown in Fig. 2, right panel. Instead, we simply set the initial state vector $\mathbf{h}^{(2)}$ to be equal to the data, $\mathbf{v}$. Using mean-field reconstructions for $\mathbf{v}$ and $\mathbf{h}^{(2)}$, one-step CD is exactly equivalent to training a modified "RBM" with only one hidden layer but with bottom-up weights that are twice the top-down weights, as defined in the original pretraining algorithm (see Fig. 1, right panel). This way of training the simple DBM with tied weights is unlikely to maximize the likelihood objective, but in practice it produces surprisingly good models that reconstruct the training data well.

When learning the second RBM in the stack, instead of maintaining a set of replicated hidden groups, it will often be convenient to approximate CD learning by training a modified RBM with one hidden layer but with asymmetric bottom-up and top-down weights.

For example, consider pretraining a two-layer DBM, in which we would like to split the modelling work between the 1$^{\text{st}}$ and 2$^{\text{nd}}$-layer RBMs as $1/3$ and $2/3$. In this case, we train the first layer RBM using one-step CD, but with the bottom-up weights constrained to be three times the top-down weights (see Fig. 3, right panel). The conditional distributions needed for CD learning take form:

$$P(h_j^{(1)} = 1|\mathbf{v}) = \frac{1}{1 + \exp(-\sum_i 3W_{ij}^{(1)} v_i)}, \qquad P(v_i = 1|\mathbf{h}^{(1)}) = \frac{1}{1 + \exp(-\sum_j W_{ij}^{(1)} h_j^{(1)})}.$$

Conversely, for the second modified RBM in the stack, the top-down weights are constrained to be $3/2$ times the bottom-up weights. The conditional distributions take form:

$$P(h_l^{(2)} = 1|\mathbf{h}^{(1)}) = \frac{1}{1 + \exp(-\sum_j 2W_{jl}^{(2)} h_j^{(1)})}, \quad P(h_j^{(1)} = 1|\mathbf{h}^{(2)}) = \frac{1}{1 + \exp(-\sum_l 3W_{jl}^{(2)} h_l^{(2)})}.$$

Note that this second-layer modified RBM simply approximates the proper RBM with three sets of replicated $\mathbf{h}^{(2)}$ groups. In practice, this simple approximation works well compared to training a proper RBM, and is much easier to implement. When combining the RBMs into a two-layer DBM, we end up with $\mathbf{W}^{(1)}$ and $2\mathbf{W}^{(2)}$ in the first and second layers, each performing $1/3$ and $2/3$ of the modelling work respectively:

$$P(\mathbf{v}; \theta) = \frac{1}{\mathcal{Z}(\theta)} \sum_{\mathbf{h}^{(1)}, \mathbf{h}^{(2)}} \exp\left(\mathbf{v}^\top \mathbf{W}^{(1)} \mathbf{h}^{(1)} + \mathbf{h}^{(1)\top} 2\mathbf{W}^{(2)} \mathbf{h}^{(2)}\right). \tag{11}$$

Parameters of the entire model can be generatively fine-tuned using the combination of the mean-field algorithm and the stochastic approximation algorithm described in Sec. 2

## 4  Pretraining a Three Layer Deep Boltzmann Machine

In the previous section, we showed that provided we start with a two-layer DBM with tied weights, we can train the second-layer RBM in a way that is guaranteed to improve the variational bound. For the DBM with more than two layers, we have not been able to develop a pretraining algorithm that is guaranteed to improve a variational bound. However, results of Sec. 3 suggest that using simple modifications when pretraining a stack of RBMs would allow us to approximately control the amount of modelling work done by each layer.

**Pretraining a 3-layer DBM**

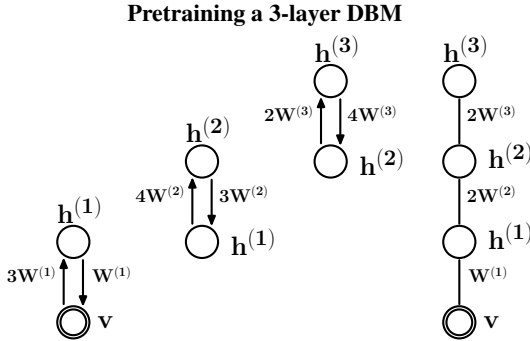

Figure 4: Layer-wise pretraining of a 3-layer Deep Boltzmann Machine.

Consider learning a 3-layer DBM, in which each layer is forced to perform approximately $1/3$ of the modelling work. This can easily be accomplished by learning a stack of three modified RBMs. Similar to the two-layer model, we train the first layer RBM using one-step CD, but with the bottom-up weights constrained to be three times the top-down weights (see Fig. 4). Two-thirds of this RBM's prior will be modelled by the 2$^{\text{nd}}$ and 3$^{\text{rd}}$-layer RBMs.

For the second modified RBM in the stack, we use $4\mathbf{W}^{(2)}$ bottom-up and $3\mathbf{W}^{(2)}$ top-down. Note that we are using $4\mathbf{W}^{(2)}$ bottom-up, as we are expecting to replace half of the second RBM prior by a third RBM, hence splitting the remaining $2/3$ of the work equally between the top two layers. If we were to pretrain only a two-layer DBM, we would use $2\mathbf{W}^{(2)}$ bottom-up and $3\mathbf{W}^{(2)}$ top-down, as discussed in Sec. 3.2.

For the last RBM in the stack, we use $2\mathbf{W}^{(3)}$ bottom-up and $4\mathbf{W}^{(2)}$ top-down. When combining the three RBMs into a three-layer DBM, we end up with symmetric weights $\mathbf{W}^{(1)}$, $2\mathbf{W}^{(2)}$, and $2\mathbf{W}^{(3)}$ in the first, second, and third layers, with each layer performing $1/3$ of the modelling work:

$$P(\mathbf{v}; \theta) = \frac{1}{\mathcal{Z}(\theta)} \sum_{\mathbf{h}} \exp\left(\mathbf{v}^\top \mathbf{W}^{(1)} \mathbf{h}^{(1)} + \mathbf{h}^{(1)\top} 2\mathbf{W}^{(2)} \mathbf{h}^{(2)} + \mathbf{h}^{(2)\top} 2\mathbf{W}^{(3)} \mathbf{h}^{(3)}\right). \tag{12}$$

**Algorithm 1** Greedy Pretraining Algorithm for a 3-layer Deep Boltzmann Machine

1: Train the $1^{st}$ layer "RBM" using one-step CD learning with mean field reconstructions of the visible vectors. Constrain the bottom-up weights, $3\mathbf{W}^{(1)}$, to be three times the top-down weights, $\mathbf{W}^{(1)}$.
2: Freeze $3\mathbf{W}^{(1)}$ that defines the $1^{st}$ layer of features, and use samples $\mathbf{h}^{(1)}$ from $P(\mathbf{h}^{(1)}|\mathbf{v}; 3\mathbf{W}^{(1)})$ as the data for training the second RBM.
3: Train the $2^{nd}$ layer "RBM" using one-step CD learning with mean field reconstructions of the visible vectors. Set the bottom-up weights to $4\mathbf{W}^{(1)}$, and the top-down weights to $3\mathbf{W}^{(1)}$.
4: Freeze $4\mathbf{W}^{(2)}$ that defines the $2^{nd}$ layer of features and use the samples $\mathbf{h}^{(3)}$ from $P(\mathbf{h}^{(2)}|\mathbf{h}^{(1)}; 4\mathbf{W}^{(2)})$ as the data for training the next RBM.
5: Train the $3^{rd}$-layer "RBM" using one-step CD learning with mean field reconstructions of its visible vectors. During the learning, set the bottom-up weights to $2\mathbf{W}^{(3)}$, and the top-down weights to $4\mathbf{W}^{(3)}$.
6: Use the weights $\{\mathbf{W}^{(1)}, 2\mathbf{W}^{(2)}, 2\mathbf{W}^{(3)}\}$ to compose a three-layer Deep Boltzmann Machine.

The new pretraining procedure for a 3-layer DBM is shown in Alg. 1. Note that compared to the original algorithm, it requires almost no extra work and can be easily integrated into existing code. Extensions to training DBMs with more layers is trivial. As we show in our experimental results, this pretraining can improve the generative performance of Deep Boltzmann Machines.

## 5    Experimental Results

In our experiments we used the MNIST and NORB datasets. During greedy pretraining, each layer was trained for 100 epochs using one-step contrastive divergence. Generative fine-tuning of the full DBM model, using mean-field together with stochastic approximation, required 300 epochs. In order to estimate the variational lower-bounds achieved by different pretraining algorithms, we need to estimate the global normalization constant. Recently, [10] demonstrated that Annealed Importance Sampling (AIS) can be used to efficiently estimate the partition function of an RBM. We adopt AIS in our experiments as well. Together with variational inference this will allow us to obtain good estimates of the lower bound on the log-probability of the training and test data.

### 5.1    MNIST

The MNIST digit dataset contains 60,000 training and 10,000 test images of ten handwritten digits (0 to 9), with $28 \times 28$ pixels. In our first experiment, we considered a standard two-layer DBM with 500 and 1000 hidden units[2], and used two different algorithms for pretraining it. The first pretraining algorithm, which we call DBM-$1/2$-$1/2$, is the original algorithm for pretraining DBMs, as introduced by [7] (see Fig. 1). Here, the modelling work between the $1^{st}$ and $2^{nd}$-layer RBMs is split equally. The second algorithm, DBM-$1/3$-$2/3$, uses a modified pretraining procedure of Sec. 3.4, so that the second RBM in the stack ends up doing $2/3$ of the modelling work compared to the $1^{st}$-layer RBM.

Results are shown in Table 1. Prior to the global generative fine-tuning, the estimate of the lower bound on the average test log-probability for DBM-$1/3$-$2/3$ was $-108.65$ per test case, compared to $-114.32$ achieved by the standard pretraining algorithm DBM-$1/2$-$1/2$. The large difference of about 7 nats shows that leaving more of the modelling work to the second layer, which has a larger number of hidden units, substantially improves the variational bound.

After the global generative fine-tuning, DBM-$1/3$-$2/3$ achieves a lower bound of $-83.43$, which is better compared to $-84.62$, achieved by DBM-$1/2$-$1/2$. This is also lower compared to the lower bound of $-85.97$, achieved by a carefully trained two-hidden-layer Deep Belief Network [10].

In our second experiment, we pretrained a 3-layer Deep Boltzmann Machine with 500, 500, and 1000 hidden units. The existing pretraining algorithm, DBM-$1/2$-$1/4$-$1/4$, approximately splits the modelling between three RBMs in the stack as $1/2$, $1/4$, $1/4$, so the weights in the $1^{st}$-layer RBM perform half of the work compared to the higher-level RBMs. On the other hand, the new pretraining procedure (see Alg. 1), which we call DBM-$1/3$-$1/3$-$1/3$, splits the modelling work equally across all three layers.

Table 1: **MNIST**: Estimating the lower bound on the average training and test log-probabilities for two DBMs: one with two layers (500 and 1000 hidden units), and the other one with three layers (500, 500, and 1000 hidden units). Results are shown for various pretraining algorithms, followed by generative fine-tuning.

|          |                              | Pretraining | | Generative Fine-Tuning | |
|----------|------------------------------|-----------|-----------|-----------|-----------|
|          |                              | Train     | Test      | Train     | Test      |
| 2 layers | DBM-$1/2$-$1/2$              | $-113.32$ | $-114.32$ | $-83.61$  | $-84.62$  |
|          | DBM-$1/3$-$2/3$              | $-107.89$ | $-108.65$ | $-82.83$  | $-83.43$  |
| 3 layers | DBM-$1/2$-$1/4$-$1/4$        | $-116.74$ | $-117.38$ | $-84.49$  | $-85.10$  |
|          | DBM-$1/3$-$1/3$-$1/3$        | $-107.12$ | $-107.65$ | $-82.34$  | $-83.02$  |

Table 2: **NORB**: Estimating the lower bound on the average training and test log-probabilities for two DBMs: one with two layers (1000 and 2000 hidden units), and the other one with three layers (1000, 1000, and 2000 hidden units). Results are shown for various pretraining algorithms, followed by generative fine-tuning.

|          |                              | Pretraining | | Generative Fine-Tuning | |
|----------|------------------------------|-----------|-----------|-----------|-----------|
|          |                              | Train     | Test      | Train     | Test      |
| 2 layers | DBM-$1/2$-$1/2$              | $-640.94$ | $-643.87$ | $-598.13$ | $-601.76$ |
|          | DBM-$1/3$-$2/3$              | $-633.21$ | $-636.65$ | $-593.76$ | $-597.23$ |
| 3 layers | DBM-$1/2$-$1/4$-$1/4$        | $-641.87$ | $-645.06$ | $-598.98$ | $-602.84$ |
|          | DBM-$1/3$-$1/3$-$1/3$        | $-632.75$ | $-635.14$ | $-592.87$ | $-596.11$ |

Table 1 shows that DBM-$1/3$-$1/3$-$1/3$ achieves a lower bound on the average test log-probability of $-107.65$, improving upon DBM-$1/2$-$1/4$-$1/4$'s bound of $-117.38$. The difference of about 10 nats further demonstrates that during the pretraining stage, it is rather crucial to push more of the modelling work to the higher layers. After generative fine-tuning, the bound on the test log-probabilities for DBM-$1/3$-$1/3$-$1/3$ was $-83.02$, so with a new pretraining procedure, the three-hidden-layer DBM performs slightly better than the two-hidden-layer DBM. With the original pretraining procedure, the 3-layer DBM achieves a bound of $-85.10$, which is worse than the bound of $84.62$, achieved by the 2-layer DBM, as reported by [7, 9].

### 5.2 NORB

The NORB dataset [4] contains images of 50 different 3D toy objects with 10 objects in each of five generic classes: cars, trucks, planes, animals, and humans. Each object is photographed from different viewpoints and under various lighting conditions. The training set contains 24,300 stereo image pairs of 25 objects, 5 per class, while the test set contains 24,300 stereo pairs of the remaining, different 25 objects. From the training data, 4,300 were set aside for validation. To deal with raw pixel data, we followed the approach of [5] by first learning a Gaussian-binary RBM with 4000 hidden units, and then treating the the activities of its hidden layer as preprocessed binary data.

Similar to the MNIST experiments, we trained two Deep Boltzmann Machines: one with two layers (1000 and 2000 hidden units), and the other one with three layers (1000, 1000, and 2000 hidden units). Table 2 reveals that for both DBMs, the new pretraining achieves much better variational bounds on the average test log-probability. Even after the global generative fine-tuning, Deep Boltzmann Machines, pretrained using a new algorithm, improve upon standard DBMs by at least 5 nats.

## 6 Conclusion

In this paper we provided a better understanding of how the pretraining algorithms for Deep Belief Networks and Deep Boltzmann Machines are related, and used this understanding to develop a different method of pretraining. Unlike many of the existing pretraining algorithms for DBNs and DBMs, the new procedure can distribute the modelling work more evenly over the hidden layers. Our results on the MNIST and NORB datasets demonstrate that the new pretraining algorithm allows us to learn much better generative models.

**Acknowledgments**
This research was funded by NSERC, Early Researcher Award, and gifts from Microsoft and Google. G.H. and R.S. are fellows of the Canadian Institute for Advanced Research.

## Footnotes

[1]We omit the bias terms for clarity of presentation.

[2]These architectures have been considered before in [7, 9], which allows us to provide a direct comparison.

## References

[1] Y. Bengio. Learning deep architectures for AI. *Foundations and Trends in Machine Learning*, 2009.

[2] G. E. Hinton, S. Osindero, and Y. W. Teh. A fast learning algorithm for deep belief nets. *Neural Computation*, 18(7):1527–1554, 2006.

[3] H. Larochelle, Y. Bengio, J. Louradour, and P. Lamblin. Exploring strategies for training deep neural networks. *Journal of Machine Learning Research*, 10:1–40, 2009.

[4] Y. LeCun, F. J. Huang, and L. Bottou. Learning methods for generic object recognition with invariance to pose and lighting. In *CVPR (2)*, pages 97–104, 2004.

[5] V. Nair and G. E. Hinton. Implicit mixtures of restricted Boltzmann machines. In *Advances in Neural Information Processing Systems*, volume 21, 2009.

[6] M. A. Ranzato. Unsupervised learning of feature hierarchies. In *Ph.D. New York University*, 2009.

[7] R. R. Salakhutdinov and G. E. Hinton. Deep Boltzmann machines. In *Proceedings of the International Conference on Artificial Intelligence and Statistics*, volume 12, 2009.

[8] R. R. Salakhutdinov and G. E. Hinton. An efficient learning procedure for Deep Boltzmann Machines. *Neural Computation*, 24:1967 – 2006, 2012.

[9] R. R. Salakhutdinov and H. Larochelle. Efficient learning of deep Boltzmann machines. In *Proceedings of the International Conference on Artificial Intelligence and Statistics*, volume 13, 2010.

[10] R. R. Salakhutdinov and I. Murray. On the quantitative analysis of deep belief networks. In *Proceedings of the International Conference on Machine Learning*, volume 25, pages 872 – 879, 2008.

[11] T. Tieleman. Training restricted Boltzmann machines using approximations to the likelihood gradient. In *ICML*. ACM, 2008.

[12] M. Welling and G. E. Hinton. A new learning algorithm for mean field Boltzmann machines. *Lecture Notes in Computer Science*, 2415, 2002.

[13] M. Welling and C. Sutton. Learning in markov random fields with contrastive free energies. In *International Workshop on AI and Statistics (AISTATS'2005)*, 2005.

[14] L. Younes. On the convergence of Markovian stochastic algorithms with rapidly decreasing ergodicity rates, March 17 2000.

[15] A. L. Yuille. The convergence of contrastive divergences. In *Advances in Neural Information Processing Systems*, 2004.

